# Extended ICA Removes Artifacts from Electroencephalographic Recordings

Tzyy-Ping Jung[1], Colin Humphries[1], Te-Won Lee[1], Scott Makeig[2,3],
Martin J. McKeown[1], Vicente Iragui[3], Terrence J. Sejnowski[1]

[1]Howard Hughes Medical Institute and Computational Neurobiology Lab
The Salk Institute, P.O. Box 85800, San Diego, CA 92186-5800
{jung,colin,tewon,scott,martin,terry}@salk.edu
[2]Naval Health Research Center, P.O. Box 85122, San Diego, CA 92186-5122
[3]Department of Neurosciences, University of California San Diego, La Jolla, CA 92093

## Abstract

Severe contamination of electroencephalographic (EEG) activity by eye movements, blinks, muscle, heart and line noise is a serious problem for EEG interpretation and analysis. Rejecting contaminated EEG segments results in a considerable loss of information and may be impractical for clinical data. Many methods have been proposed to remove eye movement and blink artifacts from EEG recordings. Often regression in the time or frequency domain is performed on simultaneous EEG and electrooculographic (EOG) recordings to derive parameters characterizing the appearance and spread of EOG artifacts in the EEG channels. However, EOG records also contain brain signals [1, 2], so regressing out EOG activity inevitably involves subtracting a portion of the relevant EEG signal from each recording as well. Regression cannot be used to remove muscle noise or line noise, since these have no reference channels. Here, we propose a new and generally applicable method for removing a wide variety of artifacts from EEG records. The method is based on an extended version of a previous Independent Component Analysis (ICA) algorithm [3, 4] for performing blind source separation on linear mixtures of independent source signals with either sub-Gaussian or super-Gaussian distributions. Our results show that ICA can effectively detect, separate and remove activity in EEG records from a wide variety of artifactual sources, with results comparing favorably to those obtained using regression-based methods.

# 1 Introduction

Eye movements, muscle noise, heart signals, and line noise often produce large and distracting artifacts in EEG recordings. Rejecting EEG segments with artifacts larger than an arbitrarily preset value is the most commonly used method for eliminating artifacts. However, when limited data are available, or blinks and muscle movements occur too frequently, as in some patient groups, the amount of data lost to artifact rejection may be unacceptable. Methods are needed for removing artifacts while preserving the essential EEG signals.

Berg & Scherg [5] have proposed a spatio-temporal dipole model for eye-artifact removal that requires *a priori* assumptions about the number of dipoles for saccade, blink, and other eye-movements, and assumes they have a simple dipolar structure. Several other proposed methods for removing eye-movement artifacts are based on regression in the time domain [6, 7] or frequency domain [8, 9]. However, simple time-domain regression tends to overcompensate for blink artifacts and may introduce *new* artifacts into EEG records [10]. The cause of this overcompensation is the difference between the spatial EOG-to-EEG transfer functions for blinks and saccades. Saccade artifacts arise from changes in orientation of the retinocorneal dipole, while blink artifacts arise from alterations in ocular conductance produced by contact of the eyelid with the cornea [11]. The transfer of blink artifacts to the recording electrodes decreases rapidly with distance from the eyes, while the transfer of saccade artifacts decreases more slowly, so that at the vertex the effect of saccades on the EEG is about double that of blinks [11], while at frontal sites the two effects may be near-equal.

Regression in the frequency domain [8, 9] can account for frequency-dependent spatial transfer function differences from EOG to EEG, but is acausal and thus unsuitable for real-time applications. Both time and frequency domain regression methods depend on having a good regressor (e.g., an EOG), and share an inherent weakness that spread of excitation from eye movements and EEG signals is bidirectional. This means that whenever regression-based artifact removal is performed, a portion of relevant EEG signals also contained in the EOG data will be cancelled out along with the eye movement artifacts. Further, since the spatial transfer functions for various EEG phenomena present in the EOG differ from the regression transfer function, their spatial distributions after artifact removal may differ from the raw record. Similar problems complicate removal of other types of EEG artifacts. Relatively little work has been done on removing muscle activity, cardiac signals and electrode noise from EEG data. Regressing out muscle noise is impractical since regressing out signals from multiple muscle groups require multiple reference channels. Line noise is most commonly filtered out in the frequency domain. However, current interest in EEG in the 40-80 Hz gamma band phenomena may make this approach undesirable as well.

We present here a new and generally applicable method for isolating and removing a wide variety of EEG artifacts by linear decomposition using a new Independent Component Analysis (ICA) algorithm [4] related to a previous algorithm [3, 12]. The ICA method is based on spatial filtering and does not rely on having a "clean" reference channel. It effectively decomposes multiple-channel EEG data into spatially-fixed and temporally independent components. Clean EEG signals can then be derived by eliminating the contributions of artifactual sources, since their time courses are generally temporally independent from and differently distributed than sources of EEG activity.

## 2   Independent Component Analysis

Bell and Sejnowski [3] have proposed a simple neural network algorithm that blindly separates mixtures, $\mathbf{x}$, of independent sources, $\mathbf{s}$, using infomax. They show that maximizing the joint entropy, $H(\mathbf{y})$, of the output of a neural processor minimizes the mutual information among the output components, $y_i = g(u_i)$, where $g(u_i)$ is an invertible bounded nonlinearity and $\mathbf{u} = \mathbf{W}\mathbf{x}$. This implies that the distribution of the output $y_i$ approximates a uniform density. Independence is achieved through the nonlinear squashing function which provides necessary higher-order statistics through its Taylor series expansion. The learning rule can be derived by maximizing output joint entropy, $H(\mathbf{y})$, with respect to $\mathbf{W}$ [3], giving,

$$\Delta\mathbf{W} \propto \frac{\partial H(\mathbf{y})}{\partial \mathbf{W}}\mathbf{W}^T\mathbf{W} = \left[\mathbf{I} + \hat{\mathbf{p}}\mathbf{u}^T\right]\mathbf{W} \tag{1}$$

where $\hat{p}_i = (\partial/\partial u_i)\ln(\partial y_i/\partial u_i)$. The 'natural gradient' $\mathbf{W}^T\mathbf{W}$ term [13] avoids matrix inversions and speeds convergence. The form of the nonlinearity $g(u)$ plays an essential role in the success of the algorithm. The ideal form for $g()$ is the cumulative density function (cdf) of the distributions of the independent sources. In practice, if we choose $g()$ to be a sigmoid function (as in [3]), the algorithm is then limited to separating sources with super-Gaussian distributions. An elegant way of generalizing the learning rule to sources with either sub- or super-Gaussian distributions is to approximate the estimated probability density function (pdf) in the form of a $4^{th}$-order Edgeworth approximation as derived by Girolami and Fyfe [14]. For sub-Gaussians, the following approximation is possible: $\hat{p}_i = +\tanh(u_i) - u_i$. For super-Gaussians, the same approximation becomes $\hat{p}_i = -\tanh(u_i) - u_i$. The sign can be chosen for each component using its normalized kurtosis, $k_4(u_i)$, giving,

$$\Delta\mathbf{W} \propto \frac{\partial H(\mathbf{y})}{\partial \mathbf{W}}\mathbf{W}^T\mathbf{W} = \left[\mathbf{I} - \text{sign}(k_4)\tanh(\mathbf{u})\mathbf{u}^T - \mathbf{u}\mathbf{u}^T\right]\mathbf{W} \tag{2}$$

Intuitively, for super-Gaussians the $-\tanh(\mathbf{u})\mathbf{u}^T$ term is an anti-Hebbian rule that tends to minimize the variance of $\mathbf{u}$, whereas for sub-Gaussians the corresponding term is a Hebbian rule that tends to maximize its variance.

### 2.1   Applying ICA to artifact correction

The ICA algorithm is effective in performing source separation in domains where, (1) the mixing medium is linear and propagation delays are negligible, (2) the time courses of the sources are independent, and (3) the number of sources is the same as the number of sensors, meaning if we employ $N$ sensors the ICA algorithm can separate $N$ sources [3, 4, 12]. In the case of EEG signals [12], volume conduction is thought to be linear and instantaneous, hence assumption (1) is satisfied. Assumption (2) is also reasonable because the sources of eye and muscle activity, line noise, and cardiac signals are not generally time locked to the sources of EEG activity which is thought to reflect activity of cortical neurons. Assumption (3) is questionable since we do not know the effective number of statistically-independent signals contributing to the scalp EEG. However, numerical simulations have confirmed that the ICA algorithm can accurately identify the time courses of activation and the scalp topographies of relatively large and temporally-independent sources from simulated scalp recordings, even in the presence of a large number of low-level and temporally-independent source activities [16].

For EEG analysis, the rows of the input matrix $\mathbf{x}$ are the EEG signals recorded at different electrodes, the rows of the output data matrix $\mathbf{u} = \mathbf{W}\mathbf{x}$ are time courses of activation of the ICA components, and the columns of the inverse matrix, $\mathbf{W}^{-1}$, give the projection strengths of the respective components onto the scalp sensors. The

scalp topographies of the components provide evidence for their biological origin (e.g., eye activity should project mainly to frontal sites). In general, and unlike PCA, the component time courses of activation will be nonorthogonal. 'Corrected' EEG signals can then be derived as $\mathbf{x}' = (\mathbf{W})^{-1}\mathbf{u}'$, where $\mathbf{u}'$ is the matrix of activation waveforms, $\mathbf{u}$, with rows representing artifactual sources set to zero.

## 3   Methods and Materials

One EEG data set used in the analysis was collected from 20 scalp electrodes placed according to the International 10-20 System and from 2 EOG placements, all referred to the left mastoid. A second EEG data set contained 19 EEG channels (no EOG channel). Data were recorded with a sampling rate of 256 Hz. ICA decomposition was performed on 10-sec EEG epochs from each data set using Matlab 4.2c on a DEC 2100A 5/300 processor. The learning batch size was 90, and initial learning rate was 0.001. Learning rate was gradually reduced to $5 \times 10^{-6}$ during 80 training iterations requiring 6.6 min of computer time. To evaluate the relative effectiveness of ICA for artifact removal, the multiple-lag regression method of Kenemans et al. [17] was performed on the same data.

## 4   Results

### 4.1   Eye movement artifacts

Figure 1 shows a 3-sec portion of the recorded EEG time series and its ICA component activations, the scalp topographies of four selected components, and the 'corrected' EEG signals obtained by removing four selected EOG and muscle noise components from the data. The eye movement artifact at 1.8 sec in the EEG data (*left*) is isolated to ICA components 1 and 2 (*left middle*). The scalp maps (*right middle*) indicate that these two components account for the spread of EOG activity to frontal sites. After eliminating these two components and projecting the remaining components onto the scalp channels, the 'corrected' EEG data (*right*) are free of these artifacts.

Removing EOG activity from frontal channels reveals alpha activity near 8 Hz that occurred during the eye movement but was obscured by the eye movement artifact in the original EEG traces. Close inspection of the EEG records (Fig. 1b) confirms its presence in the raw data. ICA also reveals the EEG 'contamination' appearing in the EOG electrodes (*right*). By contrast, the 'corrected' EEG resulting from multiple-lag regression on this data shows no sign of 8 Hz activity at Fp1 (Fig. 1b). Here, regression was performed only when the artifact was detected (1-sec surrounding the EOG peak), since otherwise a large amount of EEG activity would also have been regressed out during periods without eye movements.

### 4.2   Muscle artifacts

Left and right temporal muscle activity in the data are concentrated in ICA components 14 and 15 (Fig. 1a, *right middle*). Removing them from the data (*right*) reveals underlying EEG activity at temporal sites T3 and T4 that had been masked by muscle activity in the raw data (*left*). The signal at T3 (Fig. 1c *left*) sums muscle activity from component 14 (*center*) and underlying EEG activity. Spectral analysis of the two records (*right*) shows a large amount of overlap between their power spectra, so bandpass filtering cannot separate them. ICA component 13 (Fig. 1a, *left middle*) reveals the presence of small periodic muscle spiking (in right frontal channels, map not shown) that is highly obscured in the original data (*left*).

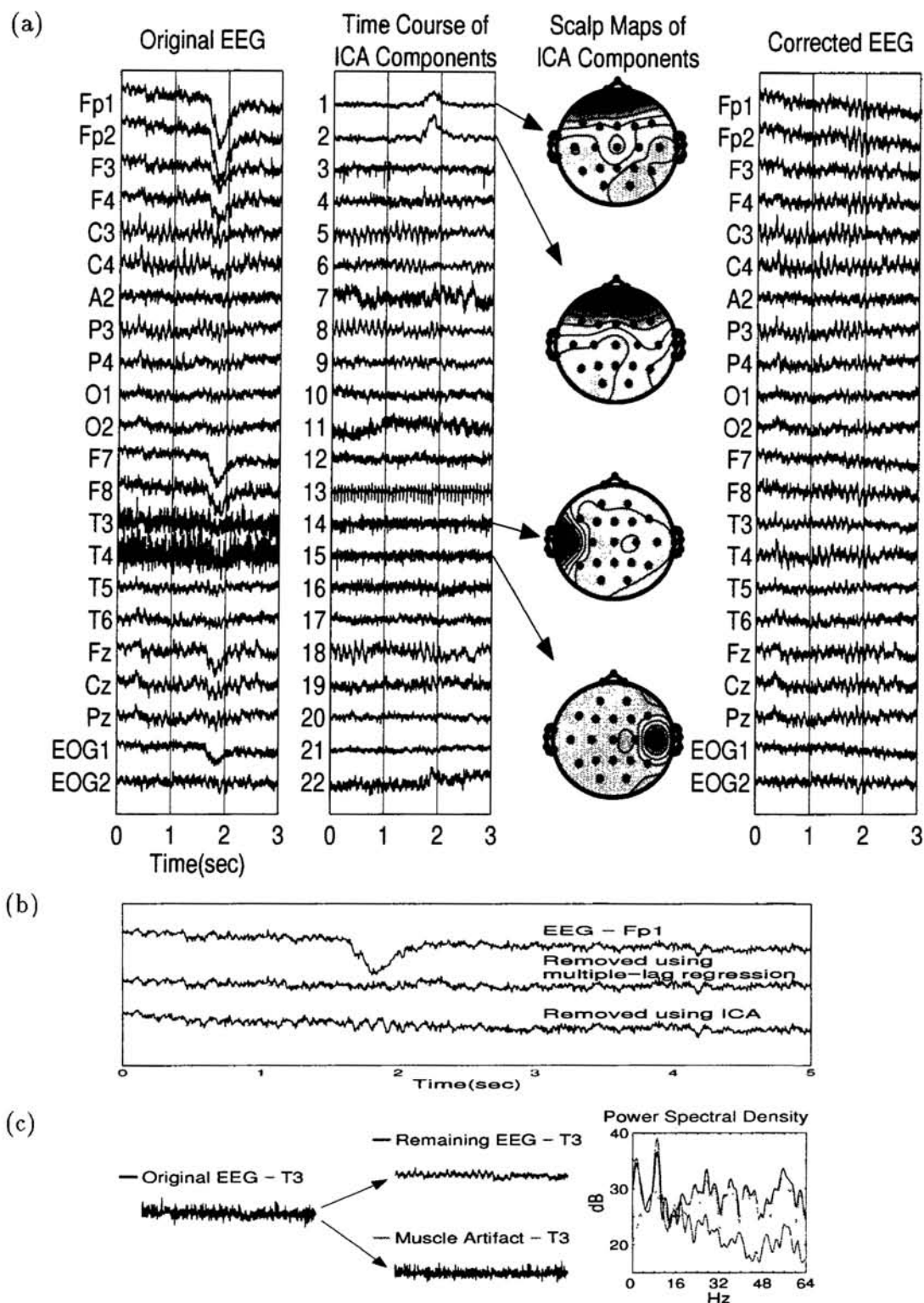

Figure 1: A 3-sec portion of an EEG time series (*left*), corresponding ICA components activations (*left middle*), scalp maps of four selected components (*right middle*), and EEG signals corrected for artifacts according to: (a) ICA with the four selected components removed (*right*), or (b) multiple-lag regression on the two EOG channels. ICA cancels multiple artifacts in all the EEG and EOG channels simultaneously. (c) The EEG record at T3 (*left*) is the sum of EEG activity recorded over the left temporal region and muscle activity occurring near the electrode (*center*). Below 20 Hz, the spectra of remaining EEG (dashed line) and muscle artifact (dotted line) overlap strongly, whereas ICA separates them by spatial filtering.

### 4.3 Cardiac contamination and line noise

Figure 2 shows a 5-sec portion of a second EEG time series, five ICA components that represent artifactual sources, and 'corrected' EEG signals obtained by removing these components. Eye blink artifacts at 0.5, 2.0 and 4.7 sec (*left*) are detected and isolated to ICA component 1 (*middle left*), even though the training data contains no EOG reference channel. The scalp map of the component captures the spread of EOG activity to frontal sites. Component 5 represents horizontal eye movements, while component 2 reveals the presence of small periodic muscle spiking in left frontal channels which is hard to see in the raw data. Line noise has a sub-Gaussian distribution and so could not be clearly isolated by earlier versions of the algorithm [3, 12]. By contrast, the new algorithm effectively concentrates the line noise present in nearly all the channels into ICA component 3. The widespread cardiac contamination in the EEG data (*left*) is concentrated in ICA component 4. After eliminating these five artifactual components, the 'corrected' EEG data (*right*) are largely free of these artifacts.

## 5  Discussion and Conclusions

ICA appears to be an effective and generally applicable method for removing known artifacts from EEG records. There are several advantages of the method: (1) ICA is computationally efficient. Although it requires more computation than the algorithm used in [15, 12], the extended ICA algorithm is effective even on large EEG data sets. (2) ICA is generally applicable to removal of a wide variety of EEG artifacts. (3) A simple analysis simultaneously separates both the EEG and its artifacts into independent components based on the statistics of the data, without relying on the availability of 'clean' reference channels. This avoids the problem of mutual contamination between regressing and regressed channels. (4) No arbitrary thresholds (variable across sessions) are needed to determine when regression should be performed. (5) Once the training is complete, artifact-free EEG records can then be derived by eliminating the contributions of the artifactual sources. However, the results of ICA are meaningful only when the amount of data and number of channels are large enough. Future work should determine the minimum data length and number of channels needed to remove artifacts of various types.

## Acknowlegement

This report was supported in part by grants from the Office of Naval Research. The views expressed in this article are those of the authors and do not reflect the official policy or position of the Department of the Navy, Department of Defense, or the U.S. Government. Dr. McKeown is supported by a grant from the Heart & Stroke Foundation of Ontario.

## References

[1] J.F. Peters (1967). Surface electrical fields generated by eye movement and eye blink potentials over the scalp, *J. EEG Technol.*, **7**:27-40.

[2] P.J. Oster & J.A. Stern (1980). Measurement of eye movement electrooculography, In: *Techniques in Psychophysiology*, Wiley, Chichester, 275-309.

[3] A.J. Bell & T.J. Sejnowski (1995). An information-maximization approach to blind separation and blind deconvolution, *Neural Computation* **7**:1129-1159.

[4] T.W. Lee and T. Sejnowski (1997). Independent Component Analysis for Sub-Gaussian and Super-Gaussian Mixtures, *Proc. 4th Joint Symp. Neural Computation* **7**:132-9.

[5] P. Berg & M. Scherg (1991) Dipole models of eye movements and blinks, *Electroencephalog. clin. Neurophysiolog.* **79**:36-44.

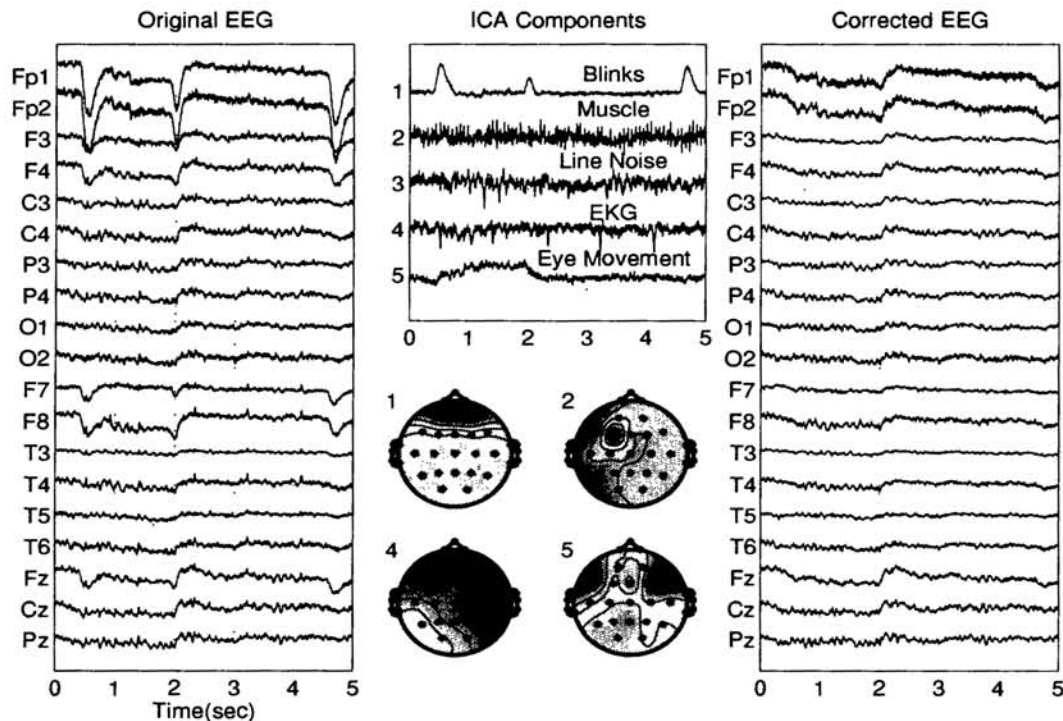

Figure 2: (*left*) A 5-sec portion of an EEG time series. (*center*) ICA components accounting for eye movements, cardiac signals, and line noise sources. (*right*) The same EEG signals 'corrected' for artifacts by removing the five selected components.

[6] S.A. Hillyard & R. Galambos (1970). Eye-movement artifact in the CNV, *Electroencephalog. clin. Neurophysiolog.* **28**:173-182.

[7] R. Verleger, T. Gasser & J. Möcks (1982). Correction of EOG artifacts in event-related potentials of EEG: Aspects of reliability and validity, *Psychoph.*, **19**(4):472-80.

[8] J.L. Whitton, F. Lue & H. Moldofsky (1978). A spectral method for removing eye-movement artifacts from the EEG. *Electroencephalog. clin. Neurophysiolog.* **44**:735-41.

[9] J.C. Woestenburg, M.N. Verbaten & J.L. Slangen (1983). The removal of the eye-movement artifact from the EEG by regression analysis in the frequency domain, *Biological Psychology* **16**:127-47.

[10] T.C. Weerts & P.J. Lang (1973). The effects of eye fixation and stimulus and response location on the contingent negative variation (CNV), *Biological Psychology* **1**(1):1-19.

[11] D.A. Overton & C. Shagass (1969). Distribution of eye movement and eye blink potentials over the scalp, *Electroencephalog. clin. Neurophysiolog.* **27**:546.

[12] S. Makeig, A.J. Bell, T.-P Jung, T.J. Sejnowski (1996) Independent Component Analysis of Electroencephalographic Data, In: *Advances in Neural Information Processing Systems* **8**:145-51.

[13] S. Amari, A. Cichocki & H. Yang (1996) A new learning algorithm for blind signal separation, In: *Advances in Neural Information Processing Systems*, **8**:757-63.

[14] M Girolami & C Fyfe (1997) Generalized Independent Component Analysis through Unsupervised Learning with Emergent Bussgang Properties, in *Proc. IEEE International Conference on Neural Networks*, 1788-91.

[15] A.J. Bell & T.J. Sejnowski (1995). Fast blind separation based on information theory, in *Proc. Intern. Symp. on Nonlinear Theory and Applications (NOLTA)* **1**:43-7.

[16] S. Makeig, T.-P Jung, D. Ghahremani & T.J. Sejnowski (1996). *Independent Component Analysis of Simulated ERP Data*, Tech. Rep. INC-9606, Institute for Neural Computation, San Diego, CA.

[17] J.L. Kenemans, P. Molenaar, M.N. Verbaten & J.L. Slangen (1991). Removal of the ocular artifact from the EEG: a comparison of time and frequency domain methods with simulated and real data, *Psychoph.*, **28**(1):114-21.
